# The Statistical Mechanics of $k$-Satisfaction

**Scott Kirkpatrick***
Racah Institute for Physics and
Center for Neural Computation
Hebrew University
Jerusalem, 91904 Israel
kirk@fiz.huji.ac.il

**Géza Györgyi**
Institute for Theoretical Physics
Eötvös University
1-1088 Puskin u. 5-7
Budapest, Hungary
gyorgyi@ludens.elte.hu,

**Naftali Tishby** and **Lidror Troyansky**
Institute of Computer Science and Center for Neural Computation
The Hebrew University of Jerusalem
91904 Jerusalem, Israel
{tishby, lidrort}@cs.huji.ac.il

## Abstract

The satisfiability of random CNF formulæ with precisely $k$ variables per clause ("$k$-SAT") is a popular testbed for the performance of search algorithms. Formulæ have $M$ clauses from $N$ variables, randomly negated, keeping the ratio $\alpha = M/N$ fixed. For $k = 2$, this model has been proven to have a sharp threshold at $\alpha = 1$ between formulæ which are almost aways satisfiable and formulæ which are almost never satisfiable as $N \to \infty$. Computer experiments for $k = 2$, 3, 4, 5 and 6, (carried out in collaboration with B. Selman of ATT Bell Labs), show similar threshold behavior for each value of $k$. Finite-size scaling, a theory of the critical point phenomena used in statistical physics, is shown to characterize the size dependence near the threshold. Annealed and replica-based mean field theories give a good account of the results.

*Permanent address: IBM TJ Watson Research Center, Yorktown Heights, NY 10598 USA. (kirk@watson.ibm.com) Portions of this work were done while visiting the Salk Institute, with support from the McDonnell-Pew Foundation.

# 1   Large-scale computation without a length scale

It is increasingly possible to model the natural world on a computer. Condensed matter physics has strategies to manage the complexities of such calculations, usually depending on a characteristic length. For example, molecules or atoms with finite ranged interactions can be broken down into weakly interacting smaller parts. We may also use symmetry to identify natural modes of the system as a whole. Even in the most difficult case, continuous phase transitions correlated over a wide range of scales, the renormalization group provides a way of collapsing the problem down to its "relevant" parts by providing a generator of behavior on all scales in terms of the critical point itself.

But length scales are not much help in organizing another sort of large calculation. Examples include large rule-based "expert systems" that model the particulars of complex industrial processes. Digital Equipment, for example, has used a network of three or more expert systems (originally called "R1/XCON") to check computer orders for completeness and internal consistency, to schedule production and shipping, and to aid a salesman to anticipate customers' needs. This very detailed set of tasks in 1979 required 2 programmers and 250 rules to deal with 100 parts. In the ten years described by Barker (1989), it grew 100X, employing 60 programmers and nearly 20,000 rules to deal with 30,000 part numbers. 100X in ten years is only moderate growth, and it would be valuable to understand how technical, social, and business factors have constrained it.

Many important commercial and scientific problems without length scales are ready for attack by computer modelling or automatic classification, and lie within a few decades of XCON's size. Retail industries routinely track $10^5 - 10^6$ distinct items kept in stock. Banks, credit card companies, and specialized information providers are building models of what $10^8$ Americans have bought and might want to buy next. In biology, human metabolism is currently described in terms of $> 1000$ substances coupled through $> 10,000$ reactions, and the data is doubling yearly. Similarly, amino acid sequences are known for $> 60,000$ proteins.

A deeper understanding of the computational cost of these problems of order $10^{6\pm2}$ is needed to see which are practical and how they can be simplified. We study an idealization of XCON-style resolution search, and find obvious collective effects which may be at the heart of its computational complexity.

# 2   Threshold Phenomena and Random $k$-SAT

Properties of randomly generated combinatorial structures often exhibit sharp threshold phenomena analogous to the phase transitions studied in condensed matter physics. Recently, thresholds have been observed in randomly generated Boolean formulae. Mitchell *et al.* (1992) consider the $k$-satisfiability problem ($k$-SAT). An instance of $k$-SAT is a Boolean formula in conjunctive normal form (CNF), *i.e.*, a conjunction (logical AND) of disjunctions or clauses (logical ORs), where each disjunction contains exactly $k$ literals. A literal is a Boolean variable or, with equal probability, its negation. The task is to determine whether there is an assignment to the variables such that all clauses evaluate to true. Here, we will use $N$ to denote the number of variables and $M$ for the number of clauses in a formula.

For randomly generated 2-SAT instances, it has been shown analytically that for large $N$, when the ratio $\alpha = M/N$ is less than 1 the instances are almost all satisfiable, whereas for ratios larger than 1, almost all instances are unsatisfiable (Chvátal and Reed 1992; Goerdt 1992). For $k \geq 3$, a rigorous analysis has proven to be elusive. Experimental evidence, however, strongly suggests a threshold with $\alpha \approx 4.3$ for 3SAT (Mitchell *et al.* 1992; Crawford and Auton 1993; Larrabee 1993).

One of the main reasons for studying randomly generated 3CNF formulae is for their use in the empirical evaluation of combinatorial search algorithms. 3CNF formulae are good candidates for the evaluation of such algorithms because determining their satisfiability is an NP-complete problem. This also holds for larger values of $k$. For $k = 1$ or 2, the satisfiability problem can be solved efficiently (Aspvall *et al.* 1979). Despite the worst-case complexity, simple heuristic methods can usually determine the satisfiability of random formulae. However, computationally challenging test instances are found by generating formulae at or near the threshold (Mitchell *et al.* 1992). Cheeseman (1991) has made a similar observation of increased computational cost for heuristic search at a boundary between two distinct phases or behaviors of a combinatorial model.

We will provide a precise characterization of the $N$-dependence of the threshold phenomena for *k*-SAT with $k$ ranging from 2 to 6. We will employ finite size scaling, a method from statistical physics in which direct observation of the width of the threshold, or "critical region" of a transition is used to characterize the "universal" behavior of quantities across the entire critical region, extending the analysis to combinatorial problems in which N characterizes the size of the model observed. For discussion of the applicability of finite-size scaling to systems without a metric, see Kirkpatrick and Selman (1993).

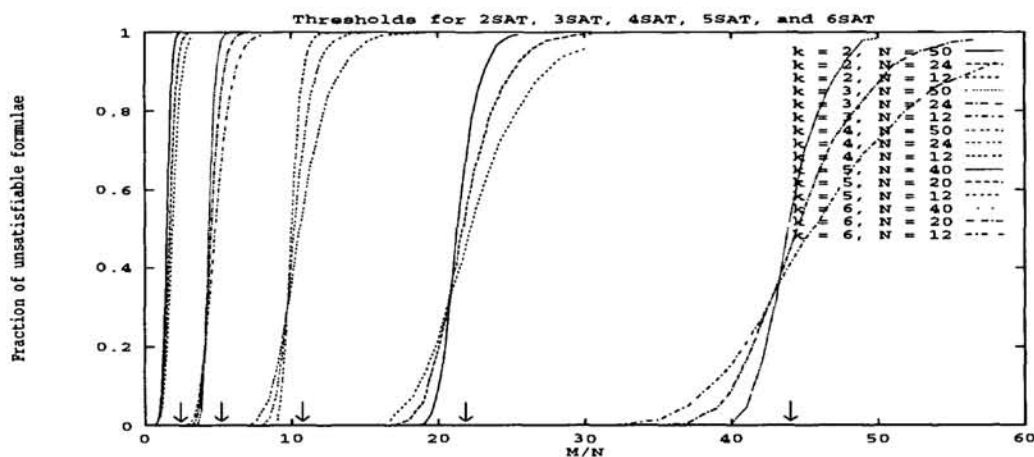

Fig. 1: Fraction of unsatisfiable formulae for 2-, 3- 4-, 5- and 6-SAT.

## 3  Experimental data

We have generated extensive data on the satisfiability of randomly generated $k$-CNF formulae with $k$ ranging from 2 to 6. Fig. 1 shows the fraction of random $k$-SAT formulae that is *unsatisfiable* as a function of the ratio, $\alpha$. For example, the left-most curve in Fig. 1 shows the fraction of formulae that is unsatisfiable for random 2CNF formulae with 50 variables over a range of values of $\alpha$.

Each data point was generated using 10000 randomly generated formulae, giving 1% accuracy. We used a highly optimized implementation of the Davis-Putnam procedure (Crawford and Auton 1993). The procedure works best on formulae with smaller $k$. Data was obtained for $k = 2$ on samples with $N \leq 500$, for $k = 3$ with $N \leq 100$, and for $k = 5$ with $N \leq 40$, all at comparable computing cost.

Fig. 1 (for $N$ ranging from 10 to 50) shows a threshold for each value of $k$. Except for the case $k = 2$, the curves cross at a single point and sharpen up with increasing $N$. For $k = 2$, the intersections between the curves for the largest values of $N$ seem to be converging to a single point as well, although the curves for smaller $N$ deviate. The point where 50% of the formulae are unsatisfiable is thought to be where the computationally hardest problems are found (Mitchell *et al.* 1992; Cheeseman *et al.* 1991). The 50% point lies consistently to the right of the scale-invariant point (the point where the curves cross each other), and shifts with $N$.

There is a simple explanation for the rapid shift of the thresholds to the right with increasing $k$. The probability that a given clause is satisfied by a random input configuration is $(2^k - 1)/2^k = (1 - 2^{-k}) \equiv \gamma_k$. If we treat the clauses as independent, the probability that all clauses are satisfied is $\gamma_k^M = \gamma_k^{\alpha N}$. We define the entropy, $S$, per input as $1/N$ times the $log_2$ of the expected number of satisfying configurations, $2^N \gamma_k^{\alpha N}$. $S = 1 + \alpha log_2(\gamma_k) \equiv 1 - \alpha/\alpha_{ann}$, and the vanishing of the entropy gives an estimate of the threshold, identical to the upper bound derived by several workers (see Franco (1983) and citations in Chvátal (1992)): $\alpha_{ann} = -(log_2(1 - 2^{-k}))^{-1} \approx (ln2)2^k$. This is called an annealed estimate for $\alpha_c$, because it ignores the interactions between clauses, just as annealed theories of materials (see Mézard 1986) average over many details of the disorder. We have marked $\alpha_{ann}$ with an arrow for each $k$ in the figures, and tabulate it in Table 1.

## 4  Results of Finite-Size Scaling Analysis

From Fig. 1, it is clear that the threshold "sharpens up" for larger values of $N$. Both the threshold shift and the increasing slope in the curves of Fig. 1 can be accounted for by finite size scaling. (See Stauffer and Aharony (1992) or Kirkpatrick and Swendsen (1985).) We plot the fraction of samples unsatisfied against the dimensionless rescaled variable,

$$y = N^{1/\nu}(\alpha - \alpha_c)/\alpha_c \ .$$

Values for $\alpha_c$ and $\nu$ must be derived from the experimental data. First $\alpha_c$ is determined as the crossing point of the curves for large $N$ in Fig. 1. Then $\nu$ is determined to make the slopes match up through the critical region. In Fig. 2 (for $k = 3$) we find that these two parameters capture both the threshold shift and the steepening of the curves, using $\alpha_c = 4.17$ and $\nu = 1.5$. We see that $F$, the fraction

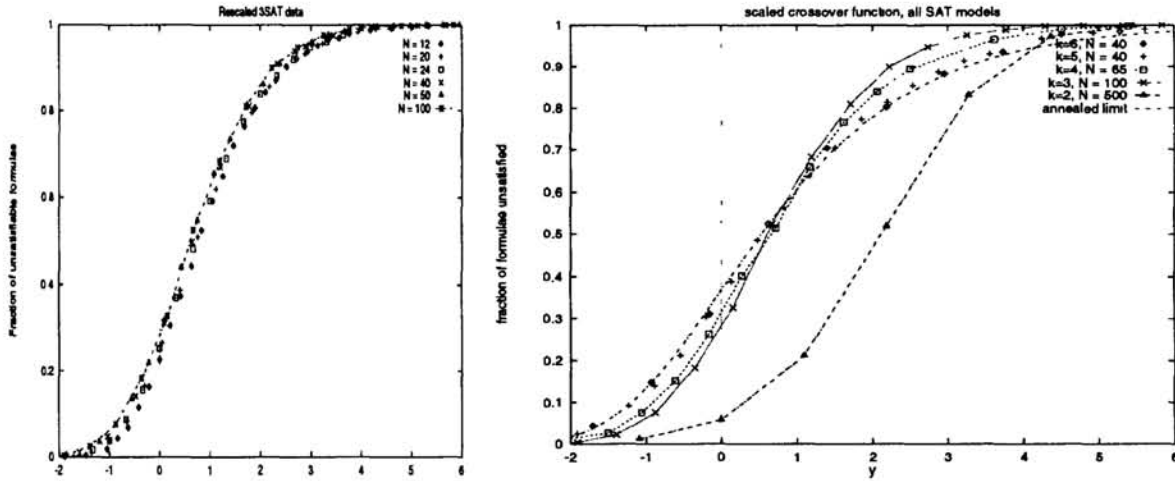

Fig. 2: Rescaled 3-SAT data using $\alpha_c = 4.17$, $\nu = 1.5$.
Fig. 3: Rescaled data for 2-, 3-, 4-, 5-, and 6-SAT approach annealed limit.

of unsatisfiable formulae, is given by $F(N, \alpha) = f(y)$, where the invariant function, $f$, is that graphed in Fig. 2.

A description of the 50% threshold shift follows immediately. If we define $y'$ by $f(y') = 0.5$, then $\alpha_{50} = \alpha_c(1 + y'N^{-1/\nu})$. From Fig. 2 we find that $\alpha_{50} \approx 4.17 + 3.1N^{-2/3}$. Crawford and Auton (1993) fit their data on the 50% point as a function of N by arbitrarily assuming that the leading correction will be $O(1/N)$. They obtain $\alpha_{50} = 4.24 + 6/N$. However, the two expressions differ by only a few percent as $N$ ranges from 10 to $\infty$.

We also obtained good results in rescaling the data for the other values of $k$. In Table 1 we give the critical parameters obtained from this analysis. The error bars are subjective, and show the range of each parameter over which the best fits were obtained. Note that $\nu$ appears to be tending to 1, and $\alpha_{ann}$ becomes an increasingly good approximation to $\alpha_c$ as $k$ increases. The success of finite-size scaling with different powers, $\nu$, is strong evidence for criticality, i.e., diverging correlations, even in the absence of any length.

Finally, we found that all the crossovers were similar in shape. In fact, combining the various rescaled curves in figure 3 shows that the curves for $k \geq 3$ all coincide in the vicinity of the 50% point, and tend to a limiting form, which can be obtained by extending the annealed arguments of the previous section. If we define

$$y_{ann} = N(\alpha - \alpha_{ann})/\alpha_{ann},$$

then the probability that a formula remains unsatisfied for all $2^N$ configurations is

$$f(y) = (1 - 2^{-(y_{ann}+N)})^{2^N} \approx e^{-2^{-y}}.$$

The curve for $k = 2$ is similar in form, but shifted to the right from the other ones.

| $k$ | $\alpha_{ann}$ | $\alpha_2$ | $\alpha_c$ | $\alpha'$ | $\nu$ |
|-----|---------|-------|---------|------|------------|
| 2 | 2.41 | 1.38 | 1.0 | 2.25 | 2.6±.2 |
| 3 | 5.19 | 4.25 | 4.17±.03 | 0.74 | 1.5±.1 |
| 4 | 10.74 | 9.58 | 9.75±.05 | 0.67 | 1.25±.05 |
| 5 | 21.83 | 20.6 | 20.9±.1 | 0.71 | 1.1±.05 |
| 6 | 44.01 | 42.8 | 43.2±.2 | 0.69 | 1.05±.05 |

Table 1: Critical parameters for random $k$-SAT.

## 5   Outline of Statistical Mechanics Analysis

Space permits only a sketch of our analysis of this model. Since the $N$ inputs are binary, we may represent them as a vector, $\mathbf{X}$, of Ising spins:

$$\mathbf{X} \equiv \{x_i = \pm 1\} \quad i = 1, \dots N.$$

Each random formula, $\mathcal{F}$, can be written as a sum of its $M$ clauses, $C_j$,

$$\mathcal{F} = \sum_{j=1}^{M} C_j,$$

where

$$C_j = \prod_{l=1}^{k} (1 - \mathbf{J}^{jl}\mathbf{X})/2.$$

where the vector, $\mathbf{J}^{j,l}$, has only one non-zero element, $\pm 1$, at the input which it selects. $\mathcal{F}$ evaluates to the number of clauses left unsatisfied by a particular configuration. It is natural to take the value of $\mathcal{F}$ to be the energy. The partition function,

$$Z = tr_{\{x_i\}} e^{\beta \mathcal{F}} = tr_{\{x_i\}} \prod_j e^{\beta C_j},$$

where $\beta$ is the inverse of a fictitious temperature, factors into contributions from each clause. The "annealed" approximation mentioned above consists simply of taking the trace over each subproduct individually, neglecting their interactions. In this construction, we expect both energy and entropy, $S$, to be extensive quantities, that is, proportional to $N$. Fig. 4 shows that this is indeed the case for $S(\alpha)$. The lines in Fig. 4 are the annealed predictions $S(\alpha, k) = 1 - \alpha/\alpha_{ann}$. Expressions for the energy can also be obtained from the annealed theory, and used to compare the specific heat observed in numerical experiments with the simple limit in which the clauses do not interact. This gives evidence supporting the identification of the unsatisfied phase as a spin glass. Finally, a plausible phase diagram for the spin glass-like "unsatisfied" phase is obtained by solving for $S(T) = 0$ at finite temperatures.

To perform the averaging over the random clauses correctly requires introducing replicas (see Mézard 1986), which are identical copies of the random formula, and defining $q$, the overlap between the expectation values of the spins in any two replicas, as the new order parameter. The results appear to be capable of accounting

for the difference between experiment and the annealed predictions at finite $k$. For example, an uncontrolled approximation in which we consider just two replicas gives the values of $\alpha_2$ in Table 1, and accounts rather closely for the average overlap found experimentally between pairs of lowest energy states, as shown in Fig. 5. The 2-replica theory gives $q$ as the solution of

$$\alpha(k,q) = 2k(1+q)^{k-1}(4^k - 2^{k+1} + (1-q)^k)/ln((1+q)(1-q))$$

for $q$ as a function of $\alpha$. This gives the lines in Fig 5. We defined $\alpha_2$ (in Table 1) as the point of inflection, or the maximum in the slope of $q(\alpha)$.

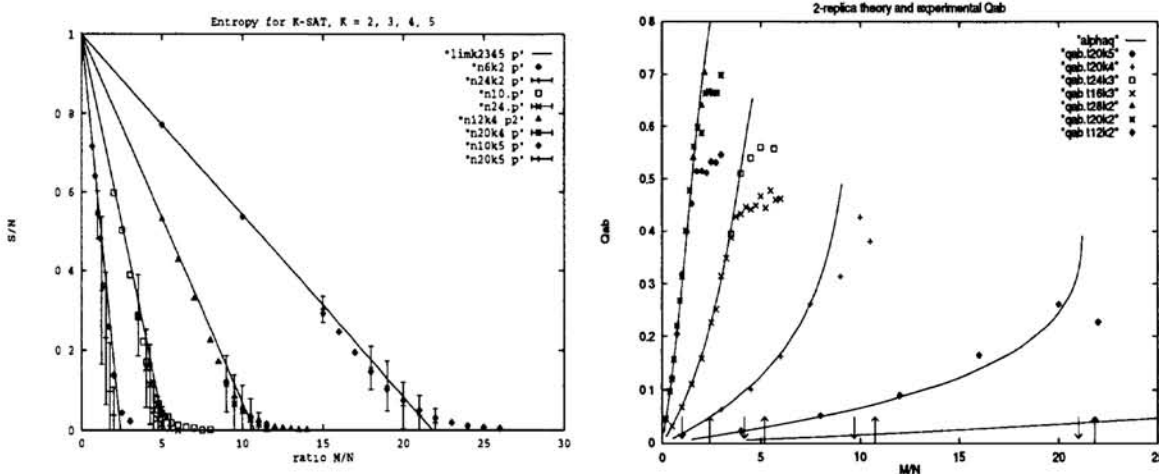

Fig. 4: Entropy as function of $\alpha$ for k = 2, 3, 4, and 5.
Fig. 5: $q$ calculated from 2-replica theory vs experimental ground state overlaps. Arrows pointing up are $\alpha_{ann}$, arrows pointing down are $\alpha_2$.

## 6    Conclusions

We have shown how finite size scaling methods from statistical physics can be used to model the threshold in randomly generated $k$-SAT problems. Given the good fit of our scaling analysis, we conjecture that this method can also give useful models of phase transitions in other combinatorial problems with a control parameter.

Several authors have attempted to relate NP-hardness or NP-completeness to the characteristics of phase transitions in models of disordered systems. Fu and Anderson (see Fu 1989) have proposed spin glasses (magnets with 2-spin interactions of random sign) as having inherent exponential complexity. Huberman and colleagues (see Clearwater 1991) were first to focus on the diverging correlation length seen at continuous phase transitions as the root of computational complexity. In fact, both effects can play important roles, but are not sufficient and may not even be necessary.

There are NP-complete problems (e.g. travelling salesman, or max-clique) which lack a phase boundary at which "hard problems" cluster. Percolation thresholds are phase transitions, yet the cost of exploring the largest cluster never exceeds N steps. Exponential search cost in $k$-SAT comes from the random signs of the inputs, which require that the space be searched repeatedly. Note that a satisfying

input configuration in 2-SAT can be determined, or its non-existence proven, in polynomial time, because it can be reduced to a percolation problem on a random directed graph (Aspvall 1979). The spin glass Hamiltonians studied by Fu and Anderson have a form close to our 2-SAT formulae, but the questions studied are different. Finding an input configuration which falsifies the minimum number of clauses is like finding the ground state in a spin glass phase, and is NP-hard when $\alpha > \alpha_c$, even for $k = 2$. Therefore, if both diverging correlations (diverging in size if no lengths are defined) and random sign or "spin-glass" effects are present, we expect a local search like Davis-Putnam to be exponentially difficult on average. But these characteristics do not imply NP-completeness.

# 7   References

Aspvall, B., Plass, M.F., and Tarjan, R.E. (1979) A linear-time algorithm for testing the truth of certain quantified Boolean formulae. *Inform. Process. Let.*, Vol. 8., 1979, 289–314.

Barker, V. E., and O'Connor, D. (1989). *Commun. Assoc. for Computing Machinery*, 32(3), 1989, 298–318.

Cheeseman, P., Kanefsky, B., and Taylor, W.M. (1991). Where the really hard problems are. *Proceedings IJCAI-91*, 1991, 163–169.

Clearwater, S.H., Huberman B.A., Hogg, T. (1991) Cooperative Solution of Constraint Satisfaction Problems. *Science*, Vol. 254, 1991, 1181-1183

Crawford, J.M. and Auton L.D. (1993). Experimental Results on the Crossover Point in Satisfiability Problems. *Proc. of AAAI-93*, 1993.

Chvátal, V. and Reed, B. (1992) Mick Gets Some: The Odds are on his Side. *Proc. of STOC*, 1992, 620–627.

Fu, Y. (1989). The Uses and Abuses of Statistical Mechanics in Computational Complexity. in *Lectures in the Sciences of Complexity*, ed. D. Stein, pp. 815-826, Addison-Wesley, 1989.

Franco, J. and Paull, M. (1988). Probabilistic Analysis of the Davis-Putnam Procedure for solving the Satisfiability Problem. *Discrete Applied Math.*, Vol. 5, 77-87, 1983.

Goerdt, A. (1992). A threshold for unsatisfiability. *Proc. 17th Int. Symp. on the Math. Foundations of Comp. Sc.*, Prague, Czechoslovakia, 1992.

Kirkpatrick, S. and Swendsen, R.H. (1985). Statistical Mechanics and Disordered Systems. *CACM*, Vol. 28, 1985, 363–373.

Kirkpatrick, S., and Selman, B. (1993), submitted for publication.

Larrabee, T. and Tsuji, Y. (1993) Evidence for a Satisfiability Threshold for Random 3CNF Formulas, *Proc. of the AAAI Spring Symposium on AI and NP-hard problems*, Palto Alto, CA, 1993.

Mézard, M., Parisi, G., Virasoro, M.A. (1986). *Spin Glass Theory and Beyond*, Singapore: World Scientific, 1986.

Mitchell, D., Selman, B., and Levesque, H.J. (1992) Hard and Easy Distributions of SAT problems. *Proc. of AAAI-92*, 1992, 456-465.

Stauffer, D. and Aharony, A. (1992) *Introduction to Percolation Theory*. London: Taylor and Francis, 1992. See especially Ch. 4.
